# Advantage Updating Applied to a Differential Game

**Mance E. Harmon**
Wright Laboratory
WL/AAAT Bldg. 635 2185 Avionics Circle
Wright-Patterson Air Force Base, OH 45433-7301
harmonme@aa.wpafb.mil

**Leemon C. Baird III\***
Wright Laboratory
baird@cs.usafa.af.mil

**A. Harry Klopf**
Wright Laboratory
klopfah@aa.wpafb.mil

**Category:** Control, Navigation, and Planning
**Keywords:** Reinforcement Learning, Advantage Updating,
Dynamic Programming, Differential Games

## Abstract

An application of reinforcement learning to a linear-quadratic, differential game is presented. The reinforcement learning system uses a recently developed algorithm, the residual gradient form of advantage updating. The game is a Markov Decision Process (MDP) with continuous time, states, and actions, linear dynamics, and a quadratic cost function. The game consists of two players, a missile and a plane; the missile pursues the plane and the plane evades the missile. The reinforcement learning algorithm for optimal control is modified for differential games in order to find the minimax point, rather than the maximum. Simulation results are compared to the optimal solution, demonstrating that the simulated reinforcement learning system converges to the optimal answer. The performance of both the residual gradient and non-residual gradient forms of advantage updating and Q-learning are compared. The results show that advantage updating converges faster than Q-learning in all simulations. The results also show advantage updating converges regardless of the time step duration; Q-learning is unable to converge as the time step duration grows small.

\* U.S.A.F. Academy, 2354 Fairchild Dr. Suite 6K41, USAFA, CO 80840-6234

# 1   ADVANTAGE UPDATING

The advantage updating algorithm (Baird, 1993) is a reinforcement learning algorithm in which two types of information are stored. For each state $x$, the value $V(x)$ is stored, representing an estimate of the total discounted return expected when starting in state $x$ and performing optimal actions. For each state $x$ and action $u$, the *advantage*, $A(x,u)$, is stored, representing an estimate of the degree to which the expected total discounted reinforcement is increased by performing action $u$ rather than the action currently considered best. The optimal value function $V^*(x)$ represents the true value of each state. The optimal advantage function $A^*(x,u)$ will be zero if $u$ is the optimal action (because $u$ confers no advantage relative to itself) and $A^*(x,u)$ will be negative for any suboptimal $u$ (because a suboptimal action has a negative advantage relative to the best action). The optimal advantage function $A^*$ can be defined in terms of the optimal value function $V^*$:

$$A^*(x,u) = \frac{1}{\Delta t}\left[R_{\Delta t}(x,u) - V^*(x) + \gamma^{\Delta t}V^*(x')\right] \tag{1}$$

The definition of an advantage includes a $1/\Delta t$ term to ensure that, for small time step duration $\Delta t$, the advantages will not all go to zero.

Both the value function and the advantage function are needed during learning, but after convergence to optimality, the policy can be extracted from the advantage function alone. The optimal policy for state $x$ is any $u$ that maximizes $A^*(x,u)$. The notation

$$A_{max}(x) = \max_u A(x,u) \tag{2}$$

defines $A_{max}(x)$. If $A_{max}$ converges to zero in every state, the advantage function is said to be *normalized*. Advantage updating has been shown to learn faster than Q-learning (Watkins, 1989), especially for continuous-time problems (Baird, 1993).

If advantage updating (Baird, 1993) is used to control a deterministic system, there are two equations that are the equivalent of the Bellman equation in value iteration (Bertsekas, 1987). These are a pair of two simultaneous equations (Baird, 1993):

$$A(x,u) - \max_{u'} A(x,u') = \left(R + \gamma^{\Delta t}V(x') - V(x)\right)\frac{1}{\Delta t} \tag{3}$$

$$\max_u A(x,u) = 0 \tag{4}$$

where a time step is of duration $\Delta t$, and performing action $u$ in state $x$ results in a reinforcement of $R$ and a transition to state $x_{t+\Delta t}$. The optimal advantage and value functions will satisfy these equations. For a given $A$ and $V$ function, the *Bellman residual errors*, $E$, as used in Williams and Baird (1993) and defined here as equations (5) and (6).are the degrees to which the two equations are not satisfied:

$$E_1(x_t,u) = \left(R(x_t,u) + \gamma^{\Delta t}V(x_{t+\Delta t}) - V(x_t)\right)\frac{1}{\Delta t} - A(x_t,u) + \max_{u'} A(x_t,u') \tag{5}$$

$$E_2(x,u) = -\max_u A(x,u) \tag{6}$$

## 2 RESIDUAL GRADIENT ALGORITHMS

Dynamic programming algorithms can be guaranteed to converge to optimality when used with look-up tables, yet be completely unstable when combined with function-approximation systems (Baird & Harmon, In preparation). It is possible to derive an algorithm that has guaranteed convergence for a quadratic function approximation system (Bradtke, 1993), but that algorithm is specific to quadratic systems. One solution to this problem is to derive a learning algorithm to perform gradient descent on the mean squared Bellman residuals given in (5) and (6). This is called the *residual gradient* form of an algorithm.

There are two Bellman residuals, (5) and (6), so the residual gradient algorithm must perform gradient descent on the sum of the two squared Bellman residuals. It has been found to be useful to combine reinforcement learning algorithms with function approximation systems (Tesauro, 1990 & 1992). If function approximation systems are used for the advantage and value functions, and if the function approximation systems are parameterized by a set of adjustable weights, and if the system being controlled is deterministic, then, for incremental learning, a given weight $W$ in the function-approximation system could be changed according to equation (7) on each time step:

$$
\begin{aligned}
\Delta W = &-\frac{\alpha}{2}\frac{\partial[E_1^2(x_t,u_t)+E_2^2(x_t,u_t)]}{\partial W} \\[2mm]
= &-\alpha E_1(x_t,u_t)\frac{\partial E_1(x_t,u_t)}{\partial W}-\alpha E_2(x_t,u_t)\frac{\partial E_2(x_t,u_t)}{\partial W} \\[2mm]
= &-\alpha\left(\frac{1}{\Delta t}\left(R+\gamma^{\Delta t}V(x_{t+\Delta t})-V(x_t)\right)-A(x_t,u_t)+\max_u A(x_t,u)\right) \\[2mm]
&\bullet\left(\frac{1}{\Delta t}\left(\gamma^{\Delta t}\frac{\partial V(x_{t+\Delta t})}{\partial W}-\frac{\partial V(x_t)}{\partial W}\right)-\frac{\partial A(x_t,u_t)}{\partial W}+\frac{\partial \max_u A(x_t,u)}{\partial W}\right) \\[2mm]
&-\alpha\max_u A(x_t,u)\frac{\partial \max_u A(x_t,u)}{\partial W}
\end{aligned}
\tag{7}
$$

As a simple, gradient-descent algorithm, equation (7) is guaranteed to converge to the correct answer for a deterministic system, in the same sense that backpropagation (Rumelhart, Hinton, Williams, 1986) is guaranteed to converge. However, if the system is nondeterministic, then it is necessary to independently generate two different possible "next states" $x_{t+\Delta t}$ for a given action $u_t$ performed in a given state $x_t$. One $x_{t+\Delta t}$ must be used to evaluate $V(x_{t+\Delta t})$, and the other must be used to evaluate $\frac{\partial}{\partial W}V(x_{t+\Delta t})$.

This ensures that the weight change is an unbiased estimator of the true Bellman-residual gradient, but requires a system such as in Dyna (Sutton, 1990) to generate the second $x_{t+\Delta t}$. The differential game in this paper was deterministic, so this was not needed here.

## 3 THE SIMULATION

### 3.1 GAME DEFINITION

We employed a linear-quadratic, differential game (Isaacs, 1965) for comparing Q-learning to advantage updating, and for comparing the algorithms in their residual gradient forms. The game has two players, a missile and a plane, as in games described by Rajan, Prasad, and Rao (1980) and Millington (1991). The state $\mathbf{x}$ is a vector $(\mathbf{x}_m, \mathbf{x}_p)$ composed of the state of the missile and the state of the plane, each of which are composed of the position and velocity of the player in two-dimensional space. The action $\mathbf{u}$ is a vector $(\mathbf{u}_m, \mathbf{u}_p)$ composed of the action performed by the missile and the action performed by the plane, each of which are the acceleration of the player in two-dimensional space. The dynamics of the system are linear; the next state $\mathbf{x}_{t+1}$ is a linear function of the current state $\mathbf{x}_t$ and action $\mathbf{u}_t$. The reinforcement function $R$ is a quadratic function of the accelerations and the distance between the players.

$$R(x,u) = [\text{distance}^2 + (\text{missile acceleration})^2 - 2(\text{plane acceleration})^2]\Delta t \tag{8}$$

$$R(\mathbf{x},\mathbf{u}) = \left[\left(\mathbf{x}_m - \mathbf{x}_p\right)^2 + \mathbf{u}_m^2 - 2\mathbf{u}_p^2\right]\Delta t \tag{9}$$

In equation (9), squaring a vector is equivalent to taking the dot product of the vector with itself. The missile seeks to minimize the reinforcement, and the plane seeks to maximize reinforcement. The plane receives twice as much punishment for acceleration as does the missile, thus allowing the missile to accelerate twice as easily as the plane.

The value function $V$ is a quadratic function of the state. In equation (10), $\mathbf{D}_m$ and $\mathbf{D}_p$ are weight matrices that change during learning.

$$V(\mathbf{x}) = \mathbf{x}_m^T \mathbf{D}_m \mathbf{x}_m + \mathbf{x}_p^T \mathbf{D}_p \mathbf{x}_p \tag{10}$$

The advantage function $A$ is a quadratic function of the state $\mathbf{x}$ and action $\mathbf{u}$. The actions are accelerations of the missile and plane in two dimensions.

$$\begin{aligned} A(\mathbf{x},\mathbf{u}) = &\mathbf{x}_m^T \mathbf{A}_m \mathbf{x}_m + \mathbf{x}_m^T \mathbf{B}_m \mathbf{C}_m \mathbf{u}_m + \mathbf{u}_m^T \mathbf{C}_m \mathbf{u}_m + \\ &\mathbf{x}_p^T \mathbf{A}_p \mathbf{x}_p + \mathbf{x}_p^T \mathbf{B}_p \mathbf{C}_p \mathbf{u}_p + \mathbf{u}_p^T \mathbf{C}_p \mathbf{u}_p \end{aligned} \tag{11}$$

The matrices $\mathbf{A}, \mathbf{B}$, and $\mathbf{C}$ are the adjustable weights that change during learning. Equation (11) is the sum of two general quadratic functions. This would still be true if the second and fifth terms were $\mathbf{xBu}$ instead of $\mathbf{xBCu}$. The latter form was used to simplify the calculation of the policy. Using the $\mathbf{xBu}$ form, the gradient is zero when $\mathbf{u} = -\mathbf{C}^{-1}\mathbf{Bx}/2$. Using the $\mathbf{xBCu}$ form, the gradient of $A(\mathbf{x},\mathbf{u})$ with respect to $\mathbf{u}$ is zero when $\mathbf{u} = -\mathbf{Bx}/2$, which avoids the need to invert a matrix while calculating the policy.

### 3.2 THE BELLMAN RESIDUAL AND UPDATE EQUATIONS

Equations (5) and (6) define the Bellman residuals when maximizing the total discounted reinforcement for an optimal control problem; equations (12) and (13) modify the algorithm to solve differential games rather than optimal control problems.

$$E_1(x_t, u_t) = \left(R(x_t, u_t) + \gamma^{\Delta t} V(x_{t+\Delta t}) - V(x_t)\right) \frac{1}{\Delta t} - A(x_t, u_t) + minimax\ A(x_t) \quad (12)$$

$$E_2(x_t, u_t) = -minimax\ A(x_t) \quad (13)$$

The resulting weight update equation is:

$$\Delta W = -\alpha \left( \left(R + \gamma^{\Delta t} V(x_{t+\Delta t}) - V(x_t)\right) \frac{1}{\Delta t} - A(x_t, u_t) + minimax\ A(x_t)\right)$$

$$\bullet \left( \left(\gamma^{\Delta t} \frac{\partial V(x_{t+\Delta t})}{\partial W} - \frac{\partial V(x_t)}{\partial W}\right) \frac{1}{\Delta t} - \frac{\partial A(x_t, u_t)}{\partial W} + \frac{\partial minimax\ A(x_t)}{\partial W}\right) \quad (14)$$

$$-\alpha minimax\ A(x_t) \frac{\partial minimax\ A(x_t)}{\partial W}$$

For Q-learning, the residual-gradient form of the weight update equation is:

$$\Delta W = -\alpha \left(R + \gamma^{\Delta t}\ minimax\ Q(x_{t+\Delta t}) - Q(x_t, u_t)\right)$$

$$\bullet \left(\gamma^{\Delta t} \tfrac{\partial}{\partial W} minimax\ Q(x_{t+\Delta t}) - \tfrac{\partial}{\partial W} Q(x_t, u_t)\right) \quad (15)$$

## 4 RESULTS

### 4.1 RESIDUAL GRADIENT ADVANTAGE UPDATING RESULTS

The optimal weight matrices $\mathbf{A}^*$, $\mathbf{B}^*$, $\mathbf{C}^*$, and $\mathbf{D}^*$ were calculated numerically with *Mathematica* for comparison. The residual gradient form of advantage updating learned the correct policy weights, $\mathbf{B}$, to three significant digits after extensive training. Very interesting behavior was exhibited by the plane under certain initial conditions. The plane learned that in some cases it is better to turn toward the missile in the short term to increase the distance between the two in the long term. A tactic sometimes used by pilots. Figure 1 gives an example.

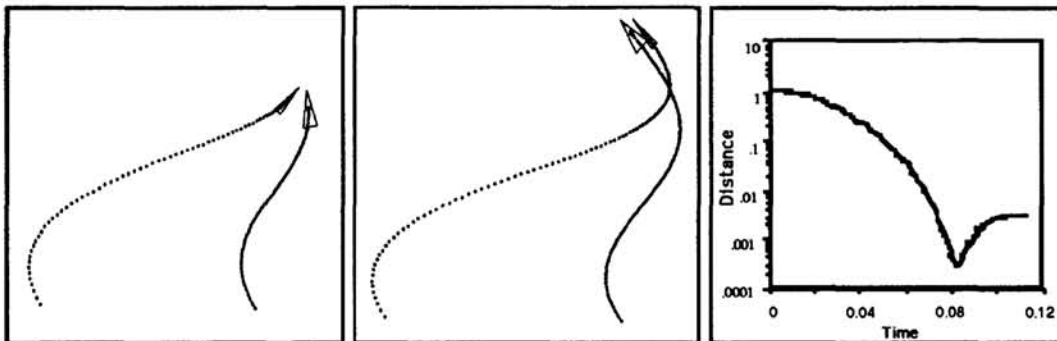

Figure 1: Simulation of a missile (dotted line) pursuing a plane (solid line), each having learned optimal behavior. The graph of distance vs. time show the effects of the plane's maneuver in turning toward the missile.

## 4.2   COMPARATIVE RESULTS

The error in the policy of a learning system was defined to be the sum of the squared errors in the **B** matrix weights. The optimal policy weights in this problem are the same for both advantage updating and Q-learning, so this metric can be used to compare results for both algorithms. Four different learning algorithms were compared: advantage updating, Q-learning, Residual Gradient advantage updating, and Residual Gradient Q-learning. Advantage updating in the non-residual-gradient form was unstable to the point that no meaningful results could be obtained, so simulation results cannot be given for it.

### 4.2.1   Experiment Set 1

The learning rates for both forms of Q-learning were optimized to one significant digit for each simulation. A single learning rate was used for residual-gradient advantage updating in all four simulations. It is possible that advantage updating would have performed better with different learning rates. For each algorithm, the error was calculated after learning for 40,000 iterations. The process was repeated 10 times using different random number seeds and the results were averaged. This experiment was performed for four different time step durations, 0.05, 0.005, 0.0005, and 0.00005. The non-residual-gradient form of Q-learning appeared to work better when the weights were initialized to small numbers. Therefore, the initial weights were chosen randomly between 0 and 1 for the residual-gradient forms of the algorithms, and between 0 and $10^{-8}$ for the non-residual-gradient form of Q-learning. For small time steps, nonresidual-gradient Q-learning performed so poorly that the error was lower for a learning rate of zero (no learning) than it was for a learning rate of $10^{-8}$. Table 1 gives the learning rates used for each simulation, and figure 2 shows the resulting error after learning.

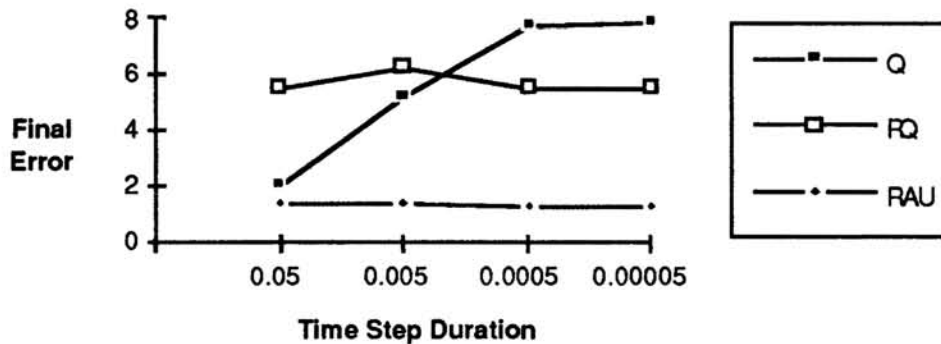

**Time Step Duration**

Figure 2: Error vs. time step size comparison for Q-Learning (Q), residual-gradient Q-Learning(RQ), and residual-gradient advantage updating(RAU) using rates optimal to one significant figure for both forms of Q-learning, and not optimized for advantage updating. The final error is the sum of squared errors in the **B** matrix weights after 40,000 time steps of learning. The final error for advantage updating was lower than both forms of Q-learning in every case. The errors increased for Q-learning as the time step size decreased.

| | Time step duration, $\Delta t$ | | | |
|---|---|---|---|---|
| | $5 \cdot 10^{-2}$ | $5 \cdot 10^{-3}$ | $5 \cdot 10^{-4}$ | $5 \cdot 10^{-5}$ |
| Q | 0.02 | 0.06 | 0.2 | 0.4 |
| RQ | 0.08 | 0.09 | 0 | 0 |
| RAU | 0.005 | 0.005 | 0.005 | 0.005 |

Table 1: Learning rates used for each simulation. Learning rates are optimal to one significant figure for both forms of Q-learning, but are not necessarily optimal for advantage updating.

### 4.2.2 Experiment Set 2

Figure 3 shows a comparison of the three algorithms' ability to converge to the correct policy. The figure shows the total squared error in each algorithms' policy weights as a function of learning time. This simulation ran for a much longer period than the simulations in table 1 and figure 2. The learning rates used for this simulation were identical to the rates that were found to be optimal for the shorter run. The weights for the non-Residual gradient form of Q-learning grew without bound in all of the long experiments, even after the learning rate was reduced by an order of magnitude. Residual gradient advantage updating was able to learn the correct policy, while Q-learning was unable to learn a policy that was better than the initial, random weights.

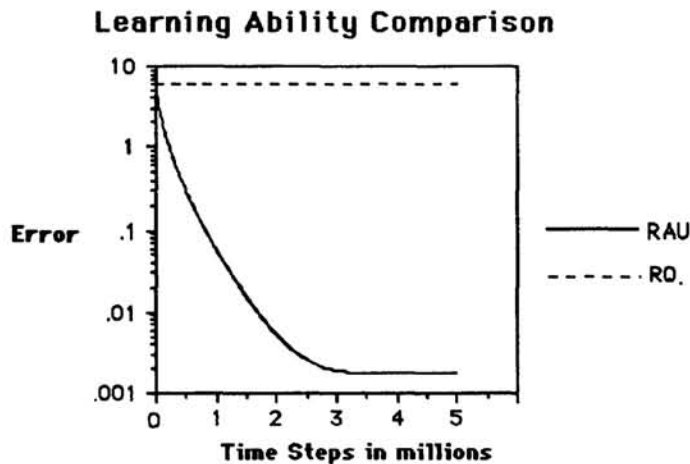

Figure 3

## 5   Conclusion

The experimental data shows residual-gradient advantage updating to be superior to the three other algorithms in all cases. As the time step grows small, Q-learning is unable to learn the correct policy. Future research will include the use of more general networks and implementation of the wire fitting algorithm proposed by Baird and Klopf (1994) to calculate the policy from a continuous choice of actions in more general networks.

**Acknowledgments**

This research was supported under Task 2312R1 by the Life and Environmental Sciences Directorate of the United States Air Force Office of Scientific Research.

**References**

Baird, L.C. (1993). *Advantage updating* Wright-Patterson Air Force Base, OH. (Wright Laboratory Technical Report WL-TR-93-1146, available from the Defense Technical information Center, Cameron Station, Alexandria, VA 22304-6145).

Baird, L.C., & Harmon, M. E. (In preparation). *Residual gradient algorithms* Wright-Patterson Air Force Base, OH. (Wright Laboratory Technical report).

Baird, L.C., & Klopf, A. H. (1993). *Reinforcement learning with high-dimensional, continuous actions* Wright-Patterson Air Force Base, OH. (Wright Laboratory technical report WL-TR-93-1147, available from the Defense Technical information Center, Cameron Station, Alexandria, VA 22304-6145).

Bertsekas, D. P. (1987). *Dynamic programming: Deterministic and stochastic models.* Englewood Cliffs, NJ: Prentice-Hall.

Bradtke, S. J. (1993). Reinforcement Learning Applied to Linear Quadratic Regulation. *Proceedings of the 5th annual Conference on Neural Information Processing Systems* .

Isaacs, Rufus (1965). *Differential games.* New York: John Wiley and Sons, Inc.

Millington, P. J. (1991). *Associative reinforcement learning for optimal control.* Unpublished master's thesis, Massachusetts Institute of Technology, Cambridge, MA.

Rajan, N., Prasad, U. R., and Rao, N. J. (1980). Pursuit-evasion of two aircraft in a horizontal plane. *Journal of Guidance and Control.* 3(3), May-June, 261-267.

Rumelhart, D., Hinton, G., & Williams, R. (1986). Learning representations by backpropagating errors. *Nature.* 323, 9 October, 533-536.

Sutton, R. S. (1990). Integrated architectures for learning, planning, and reacting based on approximating dynamic programming. *Proceedings of the Seventh International Conference on Machine Learning.*

Tesauro, G. (1990). Neurogammon: A neural-network backgammon program. *Proceedings of the International Joint Conference on Neural Networks, 3,* (pp. 33-40). San Diego, CA.

Tesauro, G. (1992). Practical issues in temporal difference learning. *Machine Learning,* 8(3/4), 279-292.

Watkins, C. J. C. H. (1989). *Learning from delayed rewards.* Doctoral thesis, Cambridge University, Cambridge, England.
